# A Winner-Take-All Circuit with Controllable Soft Max Property

**Shih-Chii Liu**
Institute for Neuroinformatics, ETH/UNIZ
Winterthurstrasse 190, CH-8057 Zurich
Switzerland
shih@ini.phys.ethz.ch

## Abstract

I describe a silicon network consisting of a group of excitatory neurons and a global inhibitory neuron. The output of the inhibitory neuron is normalized with respect to the input strengths. This output models the normalization property of the wide-field direction-selective cells in the fly visual system. This normalizing property is also useful in any system where we wish the output signal to code only the strength of the inputs, and not be dependent on the number of inputs. The circuitry in each neuron is equivalent to that in Lazzaro's winner-take-all (WTA) circuit with one additional transistor and a voltage reference. Just as in Lazzaro's circuit, the outputs of the excitatory neurons code the neuron with the largest input. The difference here is that multiple winners can be chosen. By varying the voltage reference of the neuron, the network can transition between a soft-max behavior and a hard WTA behavior. I show results from a fabricated chip of 20 neurons in a $1.2\mu$m CMOS technology.

## 1 Introduction

Lazzaro and colleagues (Lazzaro, 1988) were the first to implement a hardware model of a winner-take-all (WTA) network. This network consists of N excitatory cells that are inhibited by a global signal. Improvements of this network with addition of positive feedback and lateral connections have been described (Morris, 1998; Indiveri, 1998). The dynamics and stability properties of networks of coupled excitatory and inhibitory neurons have been analyzed by many (Amari, 1982; Grossberg, 1988). Grossberg described conditions under which these networks will exhibit WTA behavior. Lazzaro's network computes a single winner as reflected by the outputs of the excitatory cells. Several winners can be chosen by using more localized inhibition.

In this work, I describe two variants of a similar architecture where the outputs of the excitatory neurons code the relative input strengths as in a soft-max computation. The relative values of the outputs depend on the number of inputs, their relative strengths and two parameter settings in the network. The global inhibitory

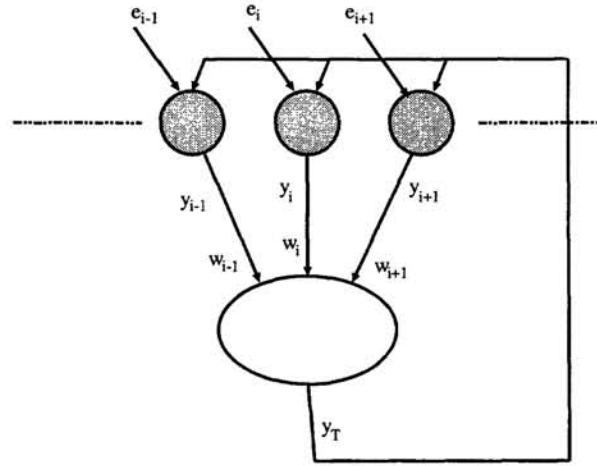

Figure 1: Network model of recurrent inhibitory network.

signal can also be used as an output. This output saturates with increasing number of active inputs, and the saturation level depends on the input strengths and parameter settings. This normalization property is similar to the normalization behavior of the wide-field direction-selective cells in the fly visual system. These cells code the temporal frequency of the visual inputs and are largely independent of the stimulation size. The circuitry in each neuron in the silicon network is equivalent to that in Lazzaro et. al.'s hard WTA network with an additional transistor and a voltage reference. By varying the voltage reference, the network can transition between a soft-max computation and a hard WTA computation. In the two variants, the outputs of the excitatory neurons either code the strength of the inputs or are normalized with respect to a constant bias current. Results from a fabricated network of 20 neurons in a $1.2\mu$m AMI CMOS show the different regimes of operation.

## 2   Network with Global Inhibition

The generic architecture of a recurrent network with excitatory neurons and a single inhibitory neuron is shown in Figure 1. The excitatory neurons receive an external input, and they synapse onto a global inhibitory neuron. The inhibitory neuron, in turn, inhibits the excitatory neurons. The dynamics of the network is described as follows:

$$\frac{dy_i}{dt} = -y_i + e_i - g(\sum_{j=1}^{N} w_j y_j) \tag{1}$$

where $w_j$ is the weight of the synapse between the $j$th excitatory neuron and the inhibitory neuron, and $y_j$ is the state of the $j$th neuron. Under steady-state conditions, $y_i = e_i - y_T$, where $y_T = g(\sum_{j=1}^{N} w_j y_j)$.

Assume a linear relationship between $y_T$ and $y_j$, and letting $w_j = w$,

$$y_T = w \sum_{j=1}^{N} y_j = \frac{w \sum_{j=1}^{N} e_j}{1 + wN}$$

As $N$ increases, $y_T = \frac{\sum_{j=1}^{N} e_j}{N}$. If all inputs have the same level, $e$, then $y_T = e$.

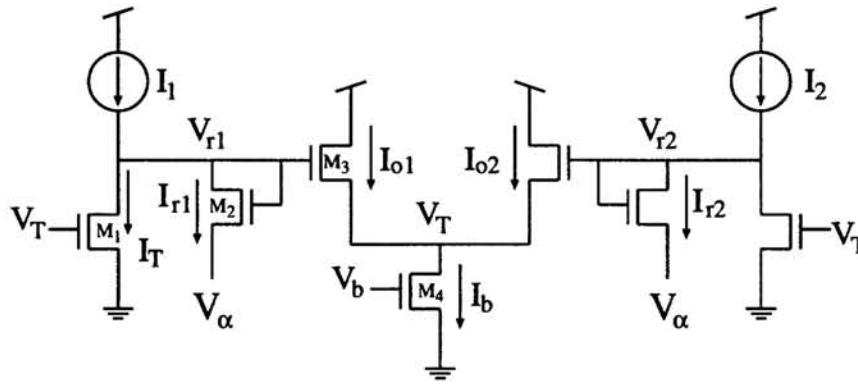

Figure 2: First variant of the architecture. Here we show the circuit for two excitatory neurons and the global inhibition neuron, $M_4$. The circuit in each excitatory neuron consists of an input current source, $I_1$, and transistors, $M_1$ to $M_3$. The inhibitory transistor is a fixed current source, $I_b$. The inputs to the inhibitory transistor, $I_{o1}$ and $I_{o2}$ are normalized with respect to $I_b$.

## 3   First Variant of Network with Fixed Current Source

In Sections 3 and 4, I describe two variants of the architecture shown in Figure 1. The two variants differ in the way that the inhibition signal is generated. The first network in Figure 2 shows the circuitry for two excitatory neurons and the inhibition neuron. Each excitatory neuron is a linear threshold unit and consists of an input current, $I_1$, and transistors, $M_1$, $M_2$, and $M_3$. The state of the neuron is represented by the current, $I_{r1}$. The diode-connected transistor, $M_2$, introduces a rectifying nonlinearity into the system since $I_{r1}$ cannot be negative. The inhibition current, $I_T$, is sunk by $M_1$, and is determined by the gate voltage, $V_T$. The inhibition neuron consists of a current source, $I_b$, and $V_T$ is determined by the corresponding current, $I_{r1}$ and the corresponding transistor, $M_3$ in each neuron. Notice that $I_T$ cannot be greater than the largest input to the network and the inputs to this network can only be excitatory. The input currents into the transistor, $M_4$, are defined as $I_{o1}$ and $I_{o2}$ and are normalized with respect to the current source, $I_b$. In the hard WTA condition, the output current of the winning neuron is equal to the bias current, $I_b$.

This network exhibits either a soft-maximum behavior or a hard WTA behavior depending on the value of an external bias, $V_\alpha$. The inhibition current, $I_T$, is derived as:

$$I_T = \frac{I_\alpha N I_i}{I_b + I_\alpha N} = \frac{N I_i}{I_b/I_\alpha + N} \tag{2}$$

where $N$ is the number of "active" excitatory neurons (that is, neurons whose $I_i > I_T$), $I_i$ is the same input current to each neuron, and $I_\alpha = I_0 e^{\kappa V_\alpha / U_T}$. In deriving the above equation, we assumed that $\kappa = 1$. The inhibition current, $I_T$, is a linear combination of the states of the neurons because $I_T = \sum_i^N I_{ri} \times I_\alpha / I_b$.

Figure 3(a) shows the response of the common-node voltage, $V_T$, as a function of the number of inputs for different input values measured from a fabricated silicon network of 20 neurons. The input current to each neuron is provided by a pFET transistor that is driven by the gate voltage, $V_{in}$. All input currents are equal in this figure. The saturation behavior of the network as a function of the number

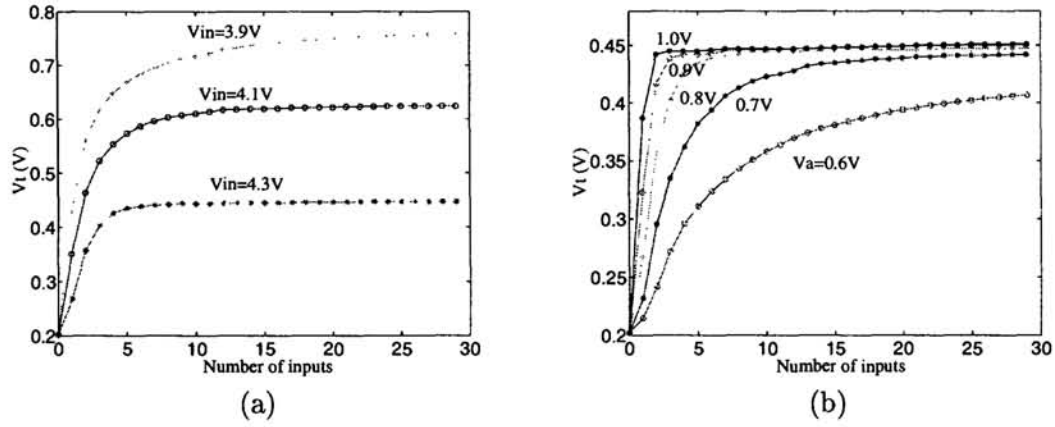

Figure 3: (a) Common-node voltage, $V_T$, as a function of the number of input stimuli. $V_\alpha = 0.8$V. (b) Common-node voltage, $V_T$, as a function of the number of inputs with an input voltage of 4.3V and $V_b = 0.7$V. The curves correspond to different values of $V_\alpha$.

of inputs can be seen in the different traces and the saturation level increases as $V_{in}$ decreases. As seen in Equation 2, the point at which the response saturates is dependent on the ratio, $I_b/I_\alpha$. In Figure 3(b), I show how the curve saturates at different points for different values of $V_\alpha$ and a fixed $I_b$ and $V_{in}$.

In Figure 4, I set all inputs to zero except for two inputs, $V_{in1}$ and $V_{in2}$ that are set to the same value. I measured $I_{o1}$ and $I_{o1}$ as a function of $V_\alpha$ as shown in Figure 4(a). The four curves correspond to four values of $V_{in}$. Initially both currents $I_{o1}$ and $I_{o2}$ are equal as is expected in the soft-max condition. As $V_\alpha$ increases, the network starts exhibiting a WTA behavior. One of the output currents finally goes to zero above a critical value of $V_\alpha$. This critical value increases for higher input currents because of transistor backgate effects. In Figure 4(b), I show how the output currents respond as a function of the differential voltage between the two inputs as shown in Figure 4. Here, I fixed one input at 4.3V and swept the second input differentially around it. The different curves correspond to different values of $V_\alpha$. For a low value of $V_\alpha$, the linear differential input range is about 100mV. This linear range decreases as $V_\alpha$ is increased (corresponding to the WTA condition).

## 4   Second Variant with Diode-Connected Inhibition Transistor

In the second variant shown in Figure 5, the current source, $M_4$ is replaced by a diode-connected transistor and the output currents, $I_{oi}$, follow the magnitude of the input currents. The inhibition current, $I_T$, can be expressed as follows:

$$I_T = (I_{ri}/I_{oi}) \times I_\alpha \tag{3}$$

where $I_\alpha$ is defined in Section 3. We sum Equation 3 over all neurons and assuming equal inputs, we get $I_T = \sqrt{\sum I_{ri}} \times I_\alpha$. This equation shows that the feedback signal has a square root dependence on the neuron states. As we will see, this causes the feedback signal to saturate quickly with the number of inputs.

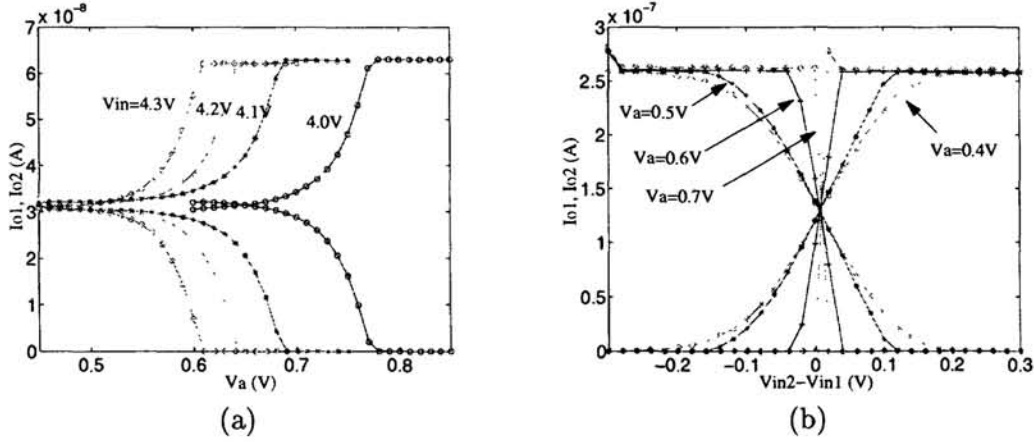

Figure 4: (a) Output currents, $I_{o1}$ and $I_{o2}$, as a function of $V_\alpha$ for a subthreshold bias current and $V_{in} = 4.0$V to 4.3V. (b) Outputs, $I_{o1}$ and $I_{o2}$, as a function of the differential input voltage, $\Delta V_{in}$, with $V_{in1} = 4.3$V.

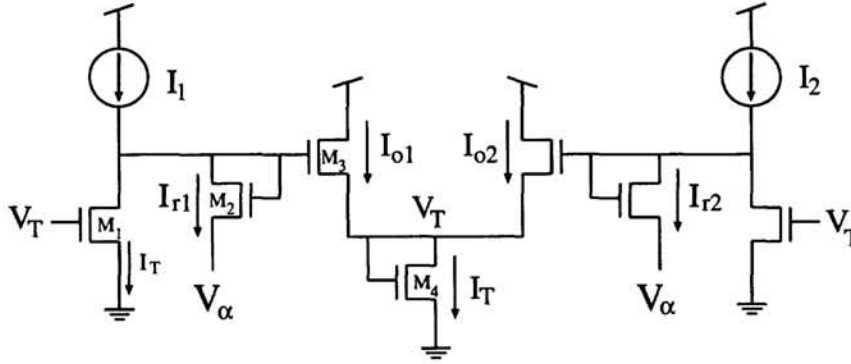

Figure 5: Second variant of network. The schematic shows two excitatory neurons with diode-connected inhibition transistor.

Substituting $I_{ri} = I_i - I_T$ in Equation 3, we solve for $I_T$,

$$I_T = -I_\alpha N + \sqrt{(I_\alpha N)^2 + 4I_\alpha \sum_i^N I_i} \tag{4}$$

From measurements from a fabricated circuit with 20 neurons, I show the dependence of $V_T$ (the natural logarithm of $I_T$) on the number of inputs in Figure 6(a). The output saturates quickly with the number of inputs and the level of saturation increases with increased input strengths. All the inputs have the same value.

The network can also act as a WTA by changing $V_\alpha$. Again, all inputs are set to zero except for two inputs whose gate voltages are both set at 4.2V. As shown in Figure 6(b), the output currents, $I_{o1}$ and $I_{o2}$, are initially equal, and as $V_\alpha$ increases above 0.6V, the output currents split apart and eventually, $I_{o2} = 0$A. The final value of $I_{o1}$ depends on the maximum input current. This data shows that the network acts as a WTA circuit when $V_\alpha > 0.73$V. If I set $V_{in2} = 4.25$V instead, the output currents split at a lower value of $V_\alpha$.

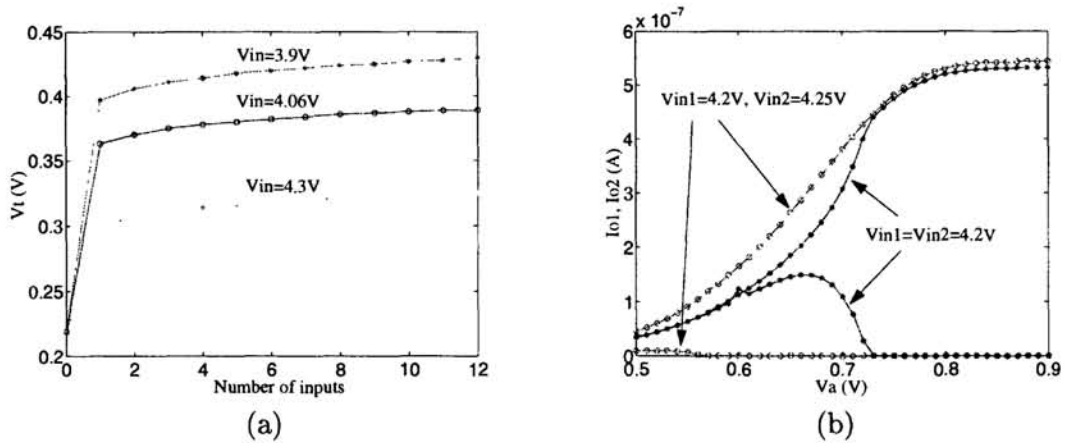

Figure 6: (a) Common-node voltage, $V_T$, as a function of the number of inputs for input voltages, 3.9V, 4.06V, and 4.3V for $V_\alpha = 0.4$V. (b) Outputs, $I_{o1}$ and $I_{o2}$, as a function of $V_\alpha$ for $V_{in1} = 4.2$V, $V_{in2} = 4.25$V for the 2 curves with asterisks and for $V_{in1} = V_{in2} = 4.2$V for the 2 curves with circles.

## 5   Inhibition

The WTA property arises in both variants of this network if the gain parameter, $V_\alpha$, is increased so that the diode-connected transistor, $M_2$, can be ignored. Both variants then reduce to Lazzaro's network. In the first variant, the feedback current ($I_T$) is a linear combination of the neuron states. However, when the gain parameter is increased so that $M_2$ can be ignored, the feedback current is now a nonlinear combination of the input states so the WTA behavior is exhibited by these reduced networks.

Under hard WTA conditions, if $I_T$ is initially smaller than all the input currents, the capacitances $C$ at the nodes $V_{r1}$ and $V_{r2}$ are charged up by the difference between the individual input current and $I_T$, i.e., $\frac{dV_{ri}}{dt} = \frac{I_i - I_T}{C}$. Since the inhibition current is a linear combination of $I_{ri}$ and $I_{ri}$ is exponential in $V_{ri}$, we can see that $I_T$ is a sum of the exponentials of the input currents, $I_i$. Hence the feedback current is nonlinear in the input currents. Another way of viewing this condition in electronic terms is that in the soft WTA condition, the output node of each neuron is a soft-impedance node, or a low-gain node. In the hard WTA case, the output node is now a high-impedance node or a high-gain node. Any input differences are immediately amplified in the circuit.

## 6   Discussion

Hahnloser (Hahnloser, 1998) recently implemented a silicon network of linear threshold excitatory neurons that are coupled to a global inhibitory neuron. The inhibitory signal is a linear combination of the output states of the excitatory neurons. This network does not exhibit WTA behavior unless the excitatory neurons include a self-excitatory term. The inhibition current in his network is also generated via a diode-connected transistor. The circuitry in two variants described here is more compact than the circuitry in his network.

Recurrent networks with the architecture described in this paper have been proposed by Reichardt and colleagues (Reichardt, 1983) in modelling the aggregation property

of the wide-field direction-selective cells in flies. The synaptic inputs are inhibited by a wide-field cell that pools all the synaptic inputs. Similar networks have also been used to model cortical processing, for example, orientation selectivity (Douglas, 1995).

The network implemented here can model the aggregation property of the direction-selective cells in the fly. By varying a voltage reference, the network implements either a soft-max computation or a hard WTA computation. This circuitry will be useful in hardware models of cortical processing or motion processing in invertebrates.

## Acknowledgments

I thank Rodney Douglas for supporting this work, and the MOSIS foundation for fabricating this circuit. I also thank Tobias Delbrück for proofreading this document. This work was supported in part by the Swiss National Foundation Research SPP grant and the U.S. Office of Naval Research.

## References

Amari, S., and Arbib, M. A., "Competition and cooperation in neural networks," New York, Springer-Verlag, 1982.

Grossberg, W., "Nonlinear neural networks: Principles, mechanisms, and architectures," *Neural Networks*, **1**, 17–61, 1988.

Hanhloser, R., "About the piecewise analysis of networks of linear threshold neurons," *Neural Networks*, **11**, 691–697, 1988.

Hahnloser, R., "Computation in recurrent networks of linear threshold neurons: Theory, simulation and hardware implementation," *Ph.D. Thesis*, Swiss Federal Institute of Technology, 1998.

Lazzaro, J., Ryckebusch, S. Mahowald, M.A., and Mead. C., "Winner-take-all networks of 0(n) complexity," In Tourestzky, D. (ed), Advances in Neural Information Processing Systems 1, San Mateo, CA: Morgan Kaufman Publishers, pp. 703–711, 1988.

Morris, T.G., Horiuchi, T. and Deweerth, S.P., "Object-based selection within an analog VLSI visual attention system," *IEEE Trans. on Circuits and Systems II*, **45:12**, 1564–1572, 1998.

Indiveri, G., "Winner-take-all networks with lateral excitation," *Neuromorphic Systems Engineering*, Editor, Lande, TS., 367–380, Kluwer Academic, Norwell, MA, 1998.

Reichardt, W., Poggio, T., and Hausen, K., "Figure-ground discrimination by relative movement in the visual system of the fly," *Biol. Cybern.*, **46**, 1–30, 1983.

Douglas, RJ., Koch, C., Mahowald, M., Martin, KAC., and Suarez, HH., "Recurrent excitation in neocortical circuits," *Science*, **269:5226**, 981–985, 1995.